# Clustering Aggregation as Maximum-Weight Independent Set

**Nan Li**      **Longin Jan Latecki**
Department of Computer and Information Sciences
Temple University, Philadelphia, USA
{nan.li,latecki}@temple.edu

## Abstract

We formulate clustering aggregation as a special instance of Maximum-Weight Independent Set (MWIS) problem. For a given dataset, an attributed graph is constructed from the union of the input clusterings generated by different underlying clustering algorithms with different parameters. The vertices, which represent the distinct clusters, are weighted by an internal index measuring both cohesion and separation. The edges connect the vertices whose corresponding clusters overlap. Intuitively, an optimal aggregated clustering can be obtained by selecting an optimal subset of non-overlapping clusters partitioning the dataset together. We formalize this intuition as the MWIS problem on the attributed graph, i.e., finding the heaviest subset of mutually non-adjacent vertices.

This MWIS problem exhibits a special structure. Since the clusters of each input clustering form a partition of the dataset, the vertices corresponding to each clustering form a maximal independent set (MIS) in the attributed graph. We propose a variant of simulated annealing method that takes advantage of this special structure. Our algorithm starts from each MIS, which is close to a distinct local optimum of the MWIS problem, and utilizes a local search heuristic to explore its neighborhood in order to find the MWIS. Extensive experiments on many challenging datasets show that: 1. our approach to clustering aggregation automatically decides the optimal number of clusters; 2. it does not require any parameter tuning for the underlying clustering algorithms; 3. it can combine the advantages of different underlying clustering algorithms to achieve superior performance; 4. it is robust against moderate or even bad input clusterings.

## 1   Introduction

Clustering is a fundamental problem in data analysis, and has extensive applications in statistics, data mining, computer vision and even in social sciences. The goal is to partition the data objects into a set of groups (clusters) such that objects in the same group are similar, while objects in different groups are dissimilar.

In the past two decades, many different clustering algorithms have been developed. Some popular ones include K-means, DBSCAN, Ward's algorithm, EM-clustering and so on. However, there are potential shortcomings for each of the known clustering algorithms. For instance, K-means [7] and its variations have difficulty detecting the "natural" clusters, which have non-spherical shapes or widely different sizes or densities. Furthermore, in order to achieve good performance, they require an appropriate number of clusters as the input parameter, which is usually very hard to specify. DBSCAN [8], a density-based clustering algorithm, can detect clusters of arbitrary shapes and sizes. However, it has trouble with data which have widely varying densities. Also, DBSCAN requires two input parameters specified by the user: the radius, $Eps$, to define the neighborhood of each data object, and the minimum number, $minPts$, of data objects required to form a cluster.

Consensus clustering, also called clustering aggregation or clustering ensemble, refers to a kind of methods which try to find a single (consensus) superior clustering from a number of input clusterings obtained by different algorithms with different parameters. The basic motivation of these methods is to combine the advantages of different clustering algorithms and overcome their respective shortcomings. Besides generating stable and robust clusterings, consensus clustering methods can be applied in many other scenarios, such as categorical data clustering, "privacy-preserving" clustering and so on. Some representative methods include [1, 2, 9, 11, 12, 13, 14]. [2] formulates clustering ensemble as a combinatorial optimization problem in terms of shared mutual information. That is, the relationship between each pair of data objects is measured based on their cluster labels from the multiple input clusterings, rather than the original features. Then a graph representation is constructed according to these relationships, and finding a single consolidated clustering is reduced to a graph partitioning problem. Similarly, in [1], a number of deterministic approximation algorithms are proposed to find an "aggregated" clustering which agrees as much as possible with the input clusterings. [9] also applies a similar idea to combine multiple runs of K-means algorithm. [11] proposes to capture the notion of agreement using an measure based on a 2D string encoding. They derive a nonlinear optimization model to maximize the new agreement measure and transform it into a strict 0-1 Semidefinite Program. [12] presents three iterative EM-like algorithms for the consensus clustering problem.

A common feature of these consensus clustering methods is that they usually do not access to the original features of the data objects. They utilize the cluster labels in different input clusterings as the new features of each data object to find an optimal clustering. Consequently, the success of these consensus clustering methods heavily relies on a premise that the majority of the input clusterings are reasonably good and consistent, which is not often the case in practice. For example, given a new challenging dataset, it is probable that only some few of the chosen underlying clustering algorithms can generate good clusterings. Many moderate or even bad input clustering can mislead the final "consensus". Furthermore, even if we choose the appropriate underlying clustering algorithms, in order to obtain good input clusterings, we still have to specify the appropriate input parameters. Therefore, it is desired to devise new consensus clustering methods which are more robust and do not need the optimal input parameters to be specified.

In this paper, our definition of "clustering aggregation" is different. Informally, for each of the clusters in the input clusterings, we evaluate its quality with some internal indices measuring both the cohesion and separation. Then we select an optimal subset of clusters, which partition the dataset together and have the best overall quality, as the "aggregated clustering". (We give a formal statement of our "clustering aggregation" problem in Sec. 2). In this framework, ideally, we can find the optimal "aggregated clustering" even if only a minority of the input clusterings are good enough. Therefore, we only need to specify an appropriate range of the input parameters, rather than the optimal values, for the underlying clustering algorithms.

We formulate this "clustering aggregation" problem as a special instance of Maximum-Weight Independent Set (MWIS) problem. An attributed graph is constructed from the union of the input clusterings. The vertices, which represent the distinct clusters, are weighted by an internal index measuring both cohesion and separation. The edges connect the vertices whose corresponding clusters overlap (In practice, we may tolerate a relatively small amount of overlap for robustness). Then selecting an optimal subset of non-overlapping clusters partitioning the dataset together can be formulated as seeking the MWIS of the attributed graph, which is the heaviest subset of mutually non-adjacent vertices. Moreover, this MWIS problem exhibits a special structure. Since the clusters of each input clustering form a partition of the dataset, the vertices corresponding to each clustering form a maximal independent set (MIS) in the attributed graph.

The most important source of motivation for our work is [3]. In [3], image segmentation is formulated as a MWIS problem. Specifically, given an image, they first segment it with different bottom-up segmentation schemes to get an ensemble of distinct superpixels. Then they select a subset of the most "meaningful" non-overlapping superpixels to partition the image. This selection procedure is formulated as solving a MWIS problem. In this respect, our work is very similar to [3]. The only difference is that our work applies the MWIS formulation to a more general problem, clustering aggregation.

MWIS problem is known to be NP-hard. Many heuristic approaches are proposed to find approximate solutions. As we mentioned before, in the context of clustering aggregation, the formulated

MWIS problem exhibits a special structure. That is, the vertices corresponding to each clustering form a maximal independent set (MIS) in the attributed graph. This special structure is valuable for finding good approximations to the MWIS because, although these MISs may not be the global optimum of the MWIS, they are close to distinct local optimums. We propose a variant of simulated annealing method that takes advantage of this special structure. Our algorithm starts from each MIS and utilizes a local search heuristic to explore its neighborhood in order to find better approximations to the MWIS. The best solution found in this process is returned as the final approximate MWIS. Since the exploration for each MIS is independent, our algorithm is suitable for parallel computation.

Finally, since the selected clusters may not be able to cover the entire dataset, our approach performs a post-processing to assign the missing data objects to their nearest clusters.

Extensive experiments on many challenging datasets show that: 1. our approach to clustering aggregation automatically decides the optimal number of clusters; 2. it does not require any parameter tuning for the underlying clustering algorithms; 3. it can combine the advantages of different underlying clustering algorithms to achieve superior performance; 4. it is robust against moderate or even bad input clusterings.

**Paper Organization** In Sec. 2, we present the formal statement of the clustering aggregation problem and its formulation as a special instance of MWIS problem. In Sec. 3, we present our algorithm. The experimental evaluations and conclusion are given in Sec. 4 and Sec. 5 respectively.

## 2 MWIS Formulation of Clustering Aggregation

Consider a set of $n$ data objects $D = \{d_1, d_2, ..., d_n\}$. A clustering $C_i$ of $D$ is obtained by applying an exclusive clustering algorithm with a specific set of input parameters on $D$. The disjoint clusters $c_{i1}, c_{i2}, ..., c_{ik}$ of $C_i$ are a partition of $D$, i.e. $\bigcup_{j=1}^{k} c_{ij} = D$ and $c_{ip} \cap c_{iq} = \emptyset$ for all $p \neq q$.

With different clustering algorithms and different parameters, we can obtain a set of $m$ different clusterings of $D$: $C_1, C_2, ..., C_m$. For each cluster $c_{ij}$ in the union of these $m$ clusterings, we evaluate its quality with an internal index measuring both cohesion and separation.

We use the average silhouette coefficient of a cluster as such an internal index in this paper. The silhouette coefficient is defined for an individual data object. It is a measure of how similar that data object is to data objects in its own cluster compared to data objects in other clusters. Formally, the silhouette coefficient for the $t^{th}$ data object, $S_t$, is defined as

$$S_t = \frac{b_t - a_t}{\max(a_t, b_t)} \tag{1}$$

where $a_t$ is the average distance from the $t^{th}$ data object to the other data objects in the same cluster as $t$, and $b_t$ is the minimum average distance from the $t^{th}$ data object to data objects in a different cluster, minimized over clusters.

Silhouette coefficient ranges from -1 to +1 and a positive value is desirable. The quality of a particular cluster $c_{ij}$ can be evaluated with the average of the silhouette coefficients of the data objects belonging to it.

$$ASC_{c_{ij}} = \frac{\sum_{t \in c_{ij}} S_t}{|c_{ij}|} \tag{2}$$

where $S_t$ is the silhouette coefficient of the $t^{th}$ data object in cluster $c_{ij}$, $|c_{ij}|$ is the cardinality of cluster $c_{ij}$.

We select an optimal subset of non-overlapping clusters from the union of all the clusterings, which partition the dataset together and have the best overall quality, as the "aggregated clustering". The selection of clusters is formulated as a special instance of the Maximum-Weight Independent Set (MWIS) problem.

Formally, consider an undirected and weighted graph $G = (V, E)$, where $V = \{1, 2, ..., n\}$ is the vertex set and $E \subseteq V \times V$ is the edge set. For each vertex $i \in V$, a positive weight $w_i$ is associated with $i$. $A = (a_{ij})_{n \times n}$ is the adjacency matrix of $G$, where $a_{ij} = 1$ if $(i, j) \in E$ is an

edge of $G$, and $a_{ij} = 0$ if $(i,j) \notin E$. A subset of $V$ can be represented by an indicator vector $\mathbf{x} = (x_i) \in \{0,1\}^n$, where $x_i = 1$ means that $i$ is in the subset, and $x_i = 0$ means that $i$ is not in the subset. An independent set is a subset of $V$, whose elements are pairwise nonadjacent. Then finding a maximum-weight independent set, denoted as $\mathbf{x}^*$ can be posed as the following:

$$\mathbf{x}^* = argmax_{\mathbf{x}}\mathbf{w^T x},$$
$$s.t. \quad \forall i \in V : x_i \in \{0,1\}, \quad \mathbf{x}^T A \mathbf{x} = 0 \tag{3}$$

The weight $w_i$ on vertex $i$ is defined as:
$$w_i = ASC_{c_i} \times |c_i| \tag{4}$$
where $c_i$ is the cluster represented by vertex $i$, $ASC_{c_i}$ and $|c_i|$ are its quality measure and cardinality respectively.

Our problem (3) is a special instance of MWIS problem, since graph $G$ exhibits an additional structure, which we will unitize in the proposed algorithm. The vertex set $V$ can be partitioned into disjoint subsets $\mathbf{P} = \{P_1, P_2, ..., P_m\}$, where $P_i$ corresponds to the clustering $C_i$, such that each $P_i$ is also a maximal independent set (MIS), which means it is not a subset of any other independent set. This follows from the fact that each clustering $C_i$ is a partition of the dataset $D$. Formally,

$$\bigcup_{i=1}^{m} P_i = V, \quad P_i \cap P_j = \emptyset, \quad i \neq j, \quad \text{and} \quad P_i \text{ is MIS}, \quad \forall i,j \in \{1,2,...,m\} \tag{5}$$

## 3 Our Algorithm

The basic idea of our algorithm is to explore the neighborhood of each known MIS $P_i$ independently with a local search heuristic in order to find better solutions. The proposed algorithm is an instance of simulated annealing methods [10] with multiple initializations.

Our algorithm starts with a particular MIS $P_i$, denoted by $x_0$. $x_{t+1}$, which is a neighbor of $x_t$, is obtained by replacing some lower-weight vertices in $x_t$ with higher-weight vertices under the constraint of always being an independent set. Specifically, we first reduce $x_t$ by removing a proportion $q$ of lower-weight vertices. Here we remove a proportion, rather than a fixed number, of vertices in order to make the reduction adaptive with respect to the number $s$ of vertices in $x_t$. In practice, we use $ceil(s \times q)$ to make sure at least one vertex will be removed. Note that this step is probabilistic, rather than deterministic. The probability that a vertex $i$ will be retained is proportional to its $WD$ value, which is defined as follows.
$$WD_i = \frac{w_i}{\sum_{j \in N_i} w_j} \tag{6}$$
where $N_i$ is the set of vertices which are connected with vertex $i$ in $G$.

Intuitively, larger $WD$ value indicates larger weight, less conflict with other vertices or both. Therefore, the obtained $x'_t$ is likely to contain vertices with large weights and have large potential room for improvement. The parameter of proportion $q$ is used to control the "radius" of the neighborhood to be explored.

Then our algorithm iteratively improves $x'_t$ by adding compatible vertices one by one. In each iteration, it first identifies all the vertices compatible with the existing ones in current $x'_t$, called candidates. Then a "local" measure $WD'$ is calculated to evaluate each of these candidates:
$$WD'_i = \frac{w_i}{\sum_{j \in N'_i} w_j} \tag{7}$$
where $N'_i$ is the set of *candidate* vertices which are connected with vertex $i$.

The large value of $WD'_i$ indicates that candidate $i$ either can bring large improvement this time (numerator) or has small conflict with further improvements (denominator) or both.

The candidate with the largest $WD'$ value is added to $x'_t$. In next iteration, this new $x'_t$ will be further improved. This iterative procedure continues until $x'_t$ cannot be further improved. We obtain $x'_t$ as a randomized neighbor of $x_t$.

**Algorithm 1:**

**Input**: Graph $G$, weights $\mathbf{w}$, adjacency matrix $\mathbf{A}$, the known MIS $P = \{P_1, P_2, ..., P_m\}$
**Output**: An approximate solution to MWIS

1 Calculate $WD$ for each vertex;
2 **for** *Each MIS $P_i$* **do**
3     Initialize $x_0$ with $P_i$;
4     **for** $t = 1, 2, ..., n$ **do**
5         Reduce $x_t$ to $x'_t$ probabilistically by removing a proportion $q$ of vertices with relatively lower $WD$ values;
6         **repeat**
7             Identify candidate vertices compatible with current $x'_t$;
8             Calculate $WD'$ for each candidate;
9             Update $x'_t$ by adding the candidate with the largest $WD'$;
10         **until** *$x'_t$ cannot be further improved*;
11         Calculate $\alpha = min[1, e^{(W(x'_t) - W(x_t))/\beta^t}]$;
12         Update $x_{t+1}$ as $x'_t$ with probability $\alpha$, otherwise $x_{t+1} = x_t$;
13     **end**
14 **end**
15 return the best solution found in the process;

Now our algorithm calculates the acceptance ratio $\alpha = e^{(W(x'_t) - W(x_t))/\beta^t}$, where $W(x) = w^T x$; $0 < \beta < 1$ is a constant which is usually picked to be close to 1. If $\alpha \geq 1$, then $x'_t$ is accepted as $x_{t+1}$. Otherwise, it is accepted with probability $\alpha$.

This exploration starting from $P_i$ continues for a number of iterations, or until $x_t$ converges. The best solution encountered in this process is recorded. After exploring the neighborhood for all the known MISs, the best solution is returned. A formal description can be found in Algorithm 1.

Our algorithm is essentially a variant of simulated annealing method [10], since the maximization of $W(x) = w^T x$ is equivalent to the minimization of the energy function $E(x) = -W(x) = -w^T x$. Lines 5 to 10 in Alg. 1 define a randomized "moving" procedure of making a transition from $x_t$ to its neighbor $x'_t$. When calculating the acceptance ratio $\alpha = e^{(W(x'_t) - W(x_t))/\beta^t}$, suppose $T_0 = 1$ (initial temperature), then it is equivalent to $\alpha = e^{(-(W(x_t) - W(x'_t)))/(\beta^t)} = e^{(-(E(x'_t) - E(x_t)))/(\beta^t)}$. Hence Algorithm 1 is a variant of simulated annealing. Therefore, our algorithm converges in theory.

In practice, the convergence of our algorithm is fast. In all the experiments presented in next section, our algorithm converges in less than 100 iterations. The reason is that our algorithm takes advantage of that the known MISs are close to distinct local maximum. Also, the local search heuristic of our algorithm is effective to find better candidate in the neighborhood.

The parameter $q$ controls the "radius" of the neighborhood to be explored in each iteration. Small $q$ means small "radius" and results in more iterations to converge. On the other side, using large $q$ will take less advantage of the known MISs. Unstable exploration also results in more iterations to converge.

Since our algorithm explores the neighborhood of each known MIS independently, its efficiency can be further improved by using parallel computation.

## 4 Results

We evaluate the performance of our approach with three experiments. In these experiments, for the underlying clustering algorithms, including K-means, single linkage, complete linkage and Ward's clustering, we use the implementations in MATLAB. Unless specified explicitly, the parameters are MATLAB's defaults. For example, when using K-means, we only specify the number $K$ of desired clusters. The default "Squared Euclidean distance" is used as the distance measure. When calculating silhouette coefficients, we use MATLAB's function "silhouette(X,clust)" and the default metric "Squared Euclidean distance". For robustness in our experiments, we tolerate slight overlap

between clusters. That is, for the adjacency matrix $A = (a_{ij})_{n \times n}$, $a_{ij} = 1$ if $\frac{|c_i \cap c_j|}{min(|c_i|,|c_j|)} > 0.1$, and $a_{ij} = 0$ otherwise. In these experiments, the parameters of our local search algorithm are: $q = 0.3$; $\beta = 0.999$; iteration number $n = 100$. We test different combinations of $q = 0.1 : 0.1 : 0.5$ and $n = 100 : 100 : 1000$. The results are almost the same.

In the first experiment, we evaluate our approach's ability to achieve good performance without specifying the optimal input parameters for the underlying clustering algorithms. We use the dataset from [6]. This dataset consists of 4 subsets (S1, S2, S3, S4) of synthetic 2-d data points. Each subset contains 5000 vectors in 15 Gaussian clusters, but with different degree of cluster overlapping. We choose K-means as the underlying clustering algorithm and vary the parameter $K = 5 : 1 : 25$, which is the desired number of clusters. Since different runs of K-means starting from random initialization of centroids typically produce different clustering results, we run K-means 5 times for each value of $K$. That is, there are a total of $21 \times 5 = 105$ different input clusterings. Note that, in order to show the performance of our approach clearly, we do not perform the post-processing of assigning the missing data points to their nearest clusters.

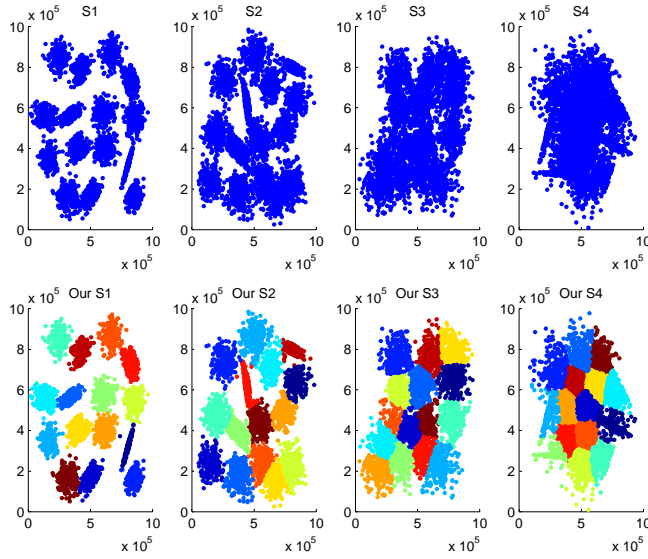

Figure 1: Clustering aggregation without parameter tuning. (top row) Original data. (bottom row) Clustering results of our approach. Best viewed in color.

As shown in Fig. 1, on each of the four subsets, the aggregated clustering obtained by our approach has the correct number (15) of clusters and near-perfect structure. Only a very small portion of data points is not assigned to any cluster. These results confirm that our approach can automatically decide the optimal number of clusters without any parameter tuning for the underlying clustering algorithms.

In the second experiment, we evaluate our approach's ability of combining the advantages of different underlying clustering algorithms and canceling out the errors introduced by them. The dataset is from [1]. As shown in the fifth panel of Fig. 2, this synthetic dataset consists of 7 distinct groups of 2-d data points, which have significantly different shapes and sizes. There are also some "bridges" between different groups of data points. Consequently, this dataset is very challenging for any single clustering algorithm. In this experiment, we use four different underlying clustering algorithms implemented in MATLAB: single linkage, complete linkage, Ward's clustering and K-means. The first two are both agglomerative bottom-up algorithms. The only difference between them is that when merging pairs of clusters, single linkage is based on the minimum distance, while complete linkage is based on maximum distance. The third one, Ward's clustering algorithm, is also an agglomerative bottom-up algorithm. In each merging step, it chooses the pair of clusters which minimize the sum of the square of distances from each point to the mean of the two clusters. The fourth algorithm is K-means.

For each of the underlying clustering algorithms, we vary the input parameter of desired number of clusters as $4 : 1 : 10$. That is, we have a total of $7 \times 4 = 28$ input clusterings.

Note that, unlike [1], we do not use the average linkage clustering algorithm, because by specifying the correct number of clusters, it can generate near-perfect clustering by itself. We abandon the best algorithm here in order to show the performance of our approach clearly. But, in practice, by utilizing good underlying clustering algorithms, it can significantly increase the chance for our approach to obtain superior aggregated clusterings. Like experiment 1, we do not perform the post-processing in this experiment.

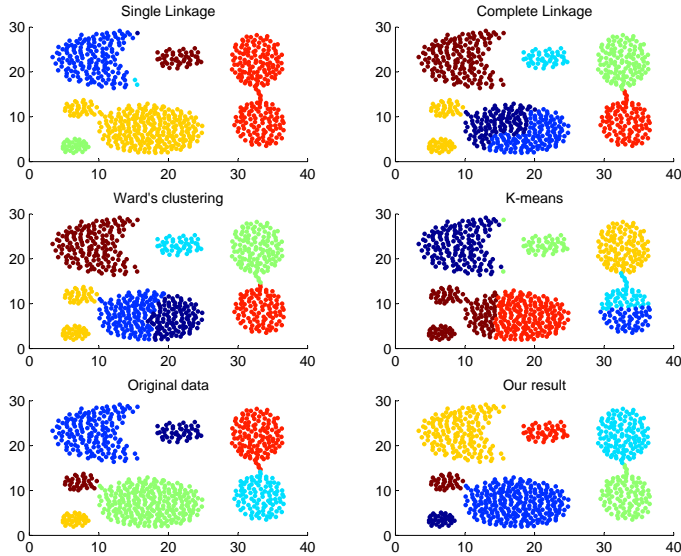

Figure 2: Clustering aggregation on four different input clusterings. Best viewed in color.

In the first four panels of Fig. 2, we show the clustering results obtained by the four underlying clustering algorithms with the number of clusters set to be 7. Obviously, even with the optimal input parameters, the results of these algorithms are far from being correct. The ground truth and the result of our approach are shown in the fifth and sixth panels, respectively. As we can see, our aggregated clustering is almost perfect, except for the three green data points in the "bridge" between the cyan and green "balls". These results confirm that our approach can effectively combine the advantages of different clustering algorithms and cancel out the errors introduced by them. Also, in contrast to the other consensus clustering algorithms, such as [1], our aggregated clustering is obtained without specifying the optimal input parameters for any of the underlying clustering algorithm. This is a very desirable feature in practice.

In the third experiment, we compare our approach with some other popular consensus clustering algorithms, including Cluster-based Similarity Partitioning Algorithm (CSPA) [2], HyperGraph Partitioning Algorithm (HGPA) [2], Meta-Clustering Algorithm (MCLA) [2], the Furthest (Furth) algorithm [1], the Agglomerative (Agglo) [1] algorithm and the Balls (Balls) algorithm [1].

The performance is evaluated on three datasets: 8D5K [2] , Iris [4] and Pen-Based Recognition of Handwritten Digits (PENDIG) [5]. 8D5K is an artificial dataset. It contains 1000 points from five multivariate Gaussian distributions (200 points each) in 8D space. Iris is a real dataset. It consists of 150 instances of three classes (50 each). There are four numeric attributes for each instance. PENDIG is also a real dataset. It contains a total of $7494 + 3498 = 10992$ instances in 10 classes. Each instance has 16 integer attributes.

For our approach and all those consensus clustering algorithms, we choose K-means and Ward's algorithm as the underlying clustering algorithms. The multiple clusterings for each dataset are obtained by varying the desired number of clusters for both K-means and Ward's algorithm. Specif-

ically, for the test on 8D5K, we set the desired numbers of clusters as 3:1:7. Consequently, there are $5 \times 2 = 10$ different input clusterings. For Iris and PENDIG, the numbers are 3:1:7 and 8:1:12 respectively. So there are also 10 different input clusterings for each of them.

In this paper, we use Jaccard coefficient to measure the quality of clusterings.

$$Jaccard\ Coefficient = \frac{f_{11}}{f_{01} + f_{10} + f_{11}} \qquad (8)$$

where $f_{11}$ is the number of object pairs which are in the same class and in the same cluster; $f_{01}$ and is the number of object pairs which are in different classes but the same cluster; $f_{10}$ is the number of object pairs which are in the same class but in different cluster.

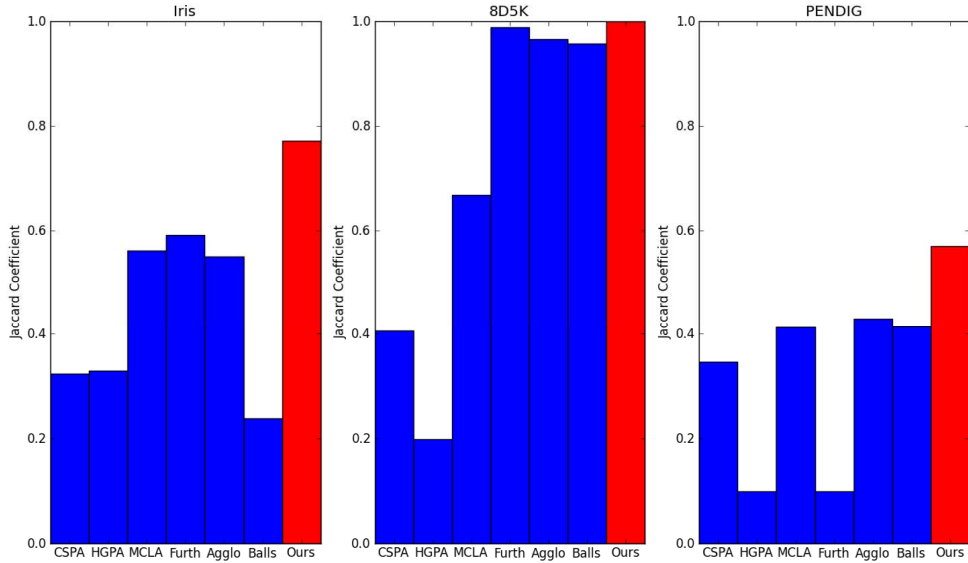

Figure 3: Results of comparative experiments on different datasets. Best viewed in color.

As shown in Fig. 3, the performance of our approach is better than those of the other consensus clustering algorithms. The main reason is that, with a range of different input parameters, most clusterings generated by the underlying clustering algorithms are not good enough. The "consensus" based on these moderate or even bad input clusterings and much less good ones cannot be good. In contrast, by selecting an optimal subset of the clusters, our approach can still achieve superior performance as long as there are good clusters in the input clusterings. Therefore, our approach is much more robust, as confirmed by the results of this experiment.

## 5   Conclusion

The contribution of this paper is twofold: 1. We formulate clustering aggregation as a MWIS problem with a special structure. 2. We propose a novel variant of simulated annealing method, which takes advantage of the special structure, for solving this special MWIS problem. Experimental results confirm that: 1. our approach to clustering aggregation automatically decides the optimal number of clusters; 2. it does not require any parameter tuning for the underlying clustering algorithms; 3. it can combine the advantages of different underlying clustering algorithms to achieve superior performance; 4. it is robust against moderate or even bad input clusterings.

### Acknowledgments

This work was supported by US Department of Energy Award 71498-001-09 and by US National Science Foundation Grants IIS-0812118, BCS-0924164, OIA-1027897.

# References

[1] Gionis, A. & Mannila, H. & Tsaparas, P. (2005) "Clustering aggregation". *Proceedings of the 21st ICDE*

[2] Strehl, A. & Ghosh, J. (2003) "Cluster ensembles—a knowledge reuse framework for combining multiple partitions". *The Journal of Machine Learning Research* (3):583-617.

[3] Brendel, W. & Todorovic, S. (2010) "Segmentation as maximum-weight independent set". *Neural Information Processing Systems*

[4] Fisher, R.A. (1936) "The use of multiple measurements in taxonomic problems". *Annual Eugenics* (7) Part II: 179-188

[5] Alimoglu, F. & Alpaydin, E. (1996) "Methods of Combining Multiple Classifiers Based on Different Representations for Pen-based Handwriting Recognition". *Proceedings of the Fifth Turkish Artificial Intelligence and Artificial Neural Networks Symposium (TAINN 96)*

[6] Franti, P. & Virmajoki, O. (2006) "Iterative shrinking method for clustering problems". *Pattern Recognition* 39 (5), 761-765

[7] Lloyd, S. P. (1982) "Least squares quantization in PCM". *IEEE Transactions on Information Theory* 28 (2): 129-137

[8] Martin Ester, Hans-Peter Kriegel, Jorg Sander, Xiaowei Xu (1996) "A density-based algorithm for discovering clusters in large spatial databases with noise". *Proceedings of the Second International Conference on Knowledge Discovery and Data Mining (KDD-96)*

[9] Fred, A.L.N. & Jain, A.K. (2002) "Data clustering using evidence accumulation". *Proceedings of the International Conference on Pattern Recognition(ICPR)* 276-280

[10] Kirkpatrick, S. & Gelatt, C. D. & Vecchi, M. P. (1983). "Optimization by Simulated Annealing". *Science* 220 (4598): 671C680

[11] Vikas Singh & Lopamudra Mukherjee & Jiming Peng & Jinhui Xu (2008) "Ensemble Clustering using Semidefinite Programming". *Advances in Neural Information Processing Systems* 20: 1353–1360

[12] Nguyen, N. & Caruana, R. (2007) "Consensus clusterings". *IEEE International Conference on Data Mining ICDM 2007* 607–612

[13] X. Z. Fern & C. E. Brodley (2004) "Solving cluster ensemble problems by bipartite graph partitioning". *Proc. of International Conference on Machine Learning* page 36

[14] Topchy, A. & Jain, A.K. & Punch, W. (2003) "Combining multiple weak clusterings". *IEEE International Conference on Data Mining, ICDM 2003* 331–338

